# Estimating Spatial Layout of Rooms using Volumetric Reasoning about Objects and Surfaces

**David C. Lee, Abhinav Gupta, Martial Hebert, Takeo Kanade**
Carnegie Mellon University
{dclee,abhinavg,hebert,tk}@cs.cmu.edu

## Abstract

There has been a recent push in extraction of 3D spatial layout of scenes. However, none of these approaches model the 3D interaction between objects and the spatial layout. In this paper, we argue for a parametric representation of objects in 3D, which allows us to incorporate volumetric constraints of the physical world. We show that augmenting current structured prediction techniques with volumetric reasoning significantly improves the performance of the state-of-the-art.

## 1 Introduction

Consider the indoor image shown in Figure 1. Understanding such a complex scene not only involves visual recognition of objects but also requires extracting the 3D spatial layout of the room (ceiling, floor and walls). Extraction of the spatial layout of a room provides crucial geometric context required for visual recognition. There has been a recent push to extract spatial layout of the room by classifiers which predict qualitative surface orientation labels (floor, ceiling, left, right, center wall and object) from appearance features and then fit a parametric model of the room. However, such an approach is limited in that it does not use the additional information conveyed by the configuration of objects in the room and, therefore, it fails to use all of the available cues for estimating the spatial layout.

In this paper, we propose to incorporate an explicit volumetric representation of objects in 3D for spatial interpretation process. Unlike previous approaches which model objects by their projection in the image plane, we propose a parametric representation of the 3D volumes occupied by objects in the scene. We show that such a parametric representation of the volume occupied by an object can provide crucial evidence for estimating the spatial layout of the rooms. This evidence comes from volumetric reasoning between the objects in the room and the spatial layout of the room. We propose to augment the existing structured classification approaches with volumetric reasoning in 3D for extracting the spatial layout of the room.

Figure 1 shows an example of a case where volumetric reasoning is crucial in estimating the surface layout of the room. Figure 1(b) shows the estimated spatial layout for the room (overlaid on surface orientation labels predicted by a classifier) when no reasoning about the objects is performed. In this case, the couch is predicted as floor and therefore there is substantial error in estimating the spatial layout. If the couch is predicted as clutter and the image evidence from the couch is ignored (Figure 1(c)), multiple room hypotheses can be selected based on the predicted labels of the pixels on the wall (Figure 1(d)) and there is still not enough evidence in the image to select one hypothesis over another in a confident manner. However, if we represent the object by a 3D parametric model, such as a cuboid (Figure 1(e)), then simple volumetric reasoning (the 3D volume occupied by the couch should be contained in the free space of the room) can help us reject physically invalid hypotheses and estimate the correct layout of the room by pushing the walls to completely contain the cuboid (Figure 1(f)).

In this paper, we propose a method to perform volumetric reasoning by combining classical constrained search techniques and current structured prediction techniques. We show that the resulting

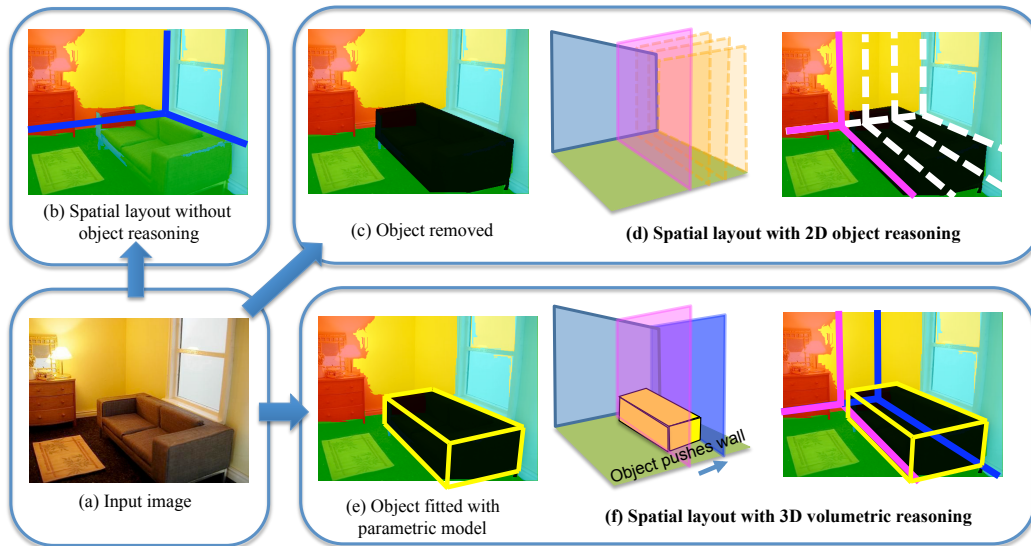

Figure 1: (a) Input image. (b) Estimate of the spatial layout of the room without object reasoning. Colors represent the output of the surface geometry by [8]. Green: floor, red: left wall, yellow: center wall, cyan: right wall. (c) Evidence from object region removed. (d) Spatial layout with 2D object reasoning. (e) Object fitted with 3D parametric model. (f) Spatial layout with 3D volumetric reasoning. The wall is pushed by the volume occupied by the object.

approach leads to substantially improved performance on standard datasets with the added benefit of a more complete scene description that includes objects in addition to surface layout.

## 1.1 Background

The goal of extracting 3D geometry by using geometric relationships between objects dates back to the start of computer vision around four decades ago. In the early days of computer vision, researchers extracted lines from "blockworld" scenes [1] and used geometric relationships using constraint satisfaction algorithms on junctions [2, 3]. However, the reasoning approaches used in these block world scenarios (synthetic line drawings) proved too brittle for the real-world images and could not handle the errors in extraction of line-segments or generalize to other shapes.

In recent years, there has been renewed interest in extracting camera parameters and three-dimensional structures in restricted domains such as *Manhattan Worlds* [4]. Kosecka et al. [5] developed a method to recover vanishing points and camera parameters from a single image by using line segments found in Manhattan structures. Using the recovered vanishing points, rectangular surfaces aligned with major orientations were also detected by [6]. However, these approaches are only concerned with dominant directions in the 3D world and do not attempt extract three dimensional information of the room and the objects in the room. Yu et al. [7] inferred the relative depth-order of rectangular surfaces by considering their relationship. However, this method only provides depth cues of partial rectangular regions in the image and not the entire scene.

There has been a recent series of methods related to our work that attempt to model geometric scene structure from a single image, including geometric label classification [8, 9] and finding vertical/ground fold-lines [10]. Lee et al. [11] introduced parameterized models of indoor environments, constrained by rules inspired by blockworld to guarantee physical validity. However, since this approach samples possible spatial layout hypothesis without clutter, it is prone to errors caused by the occlusion and tend to fit rooms in which the walls coincide with the object surfaces. A recent paper by Hedau et al. [12] uses an appearance based clutter classifier and computes visual features only from the regions classified as "non-clutter", while parameterizing the 3D structure of the scene by a box. They use structured approaches to estimate the best fitting room box to the image. A similar approach has been used by Wang et al. [13] which does not require the ground truth lables of clutter. In these methods, however, the modeling of interactions between clutter and spatial-layout of the room is only done in the image plane and the 3D interactions between room and clutter are not considered.

In a work concurrent to ours, Hedau et al. [14] have also modeled objects as three dimensional cuboids and considered the volumetric intersection with the room structure. The goal of their work differs from ours. Their primary goal is to improve object detection, such as beds, by using information of scene geometry, whereas our goal is to improve scene understanding by proposing a control structure that incorporates volumetric constraints. Therefore, we are able to improve the estimate of the room by estimating the objects and vice versa, whereas in their work information flows in only one direction (from scene to objects).

In a very recent work by Gupta et al. [15], qualitative reasoning of scene geometry was done by modeling objects as "blocks" for outdoor scenes. In contrast, we use stronger parameteric models for rooms and objects in indoor scenes, which are more structured, that allows us to do more explicit and exact 3D volumetric reasoning.

## 2 Overview

Our goal is to jointly extract the spatial layout of the room and the configuration of objects in the scene. We model the spatial layout of the room by 3D boxes and we model the objects as solids which occupy 3D volumes in the free space defined by the room walls. Given a set of room hypotheses and object hypotheses, our goal is to search the space of scene configurations and select the configuration that best matches the local surface geometry estimated from image cues and satisfies the volumetric constraints of the physical world. These constraints (shown in Figure 3(i)) are:

- **Finite volume:** Every object in the world should have a non-zero finite volume.
- **Spatial exclusion:** The objects are assumed to be solid objects which cannot intersect. Therefore, the volumes occupied by different object are mutually exclusive. This implies that the volumetric intersection between two objects should be empty.
- **Containment:** Every object should be contained in the free space defined by the walls of the room (i.e, none of the objects should be outside the room walls).

Our approach is illustrated in Figure 2. We first extract line segments and estimate three mutually orthogonal vanishing points (Figure 2(b)). The vanishing points define the orientation of the major surfaces in the scene [6, 11, 12] and hence constrain the layout of ceilings, floor and walls of the room. Using the line segments labeled by their orientations, we then generate multiple hypotheses for rooms and objects (Figure 2(e)(f)). A hypothesis of a room is a 3D parametric representation of the layout of major surfaces of the scene, such as floor, left wall, center wall, right wall, and ceiling. A hypothesis of an object is a 3D parametric representation of an object in the scene, approximated as a cuboid.

The room and cuboid hypotheses are then combined to form the set of possible configurations of the entire scene (Figure 2(h)). The configuration of the entire scene is represented as one sample of the room hypothesis along with some subset of object hypotheses. The number of possible scene configurations is exponential in the number of object hypotheses [1]. However, not all cuboid and room subsets are compatible with each other. We use simple 3D spatial reasoning to enforce the volumetric constraints described above (See Figure 2(g)). We therefore test each room-object pair and each object-object pair for their 3D volumetric compatibility, so that we allow only the scene configurations which have no room-object and no object-object volumetric intersection.

Finally, we evaluate the scene configurations created by combinations of room hypotheses and object hypotheses to find the scene configuration that best matches the image (Figure 2(i)). As the scene configuration is a structured variable, we use a variant of the structured prediction algorithm [16] to learn the cost function. We use two sources of surface geometry, orientation map [11] and geometric context [8], which serve as features in the cost function. Since it is computationally expensive to test exhaustive combinations of scene configurations in practice, we use beam-search to sample the scene configurations that are volumetrically-compatible (Section 5.1).

## 3 Estimating Surface Geometry

We would like to predict the local surface geometry of the regions in the image. A scene configuration should satisfy local surface geometry extracted from image cues and should satisfy the 3D

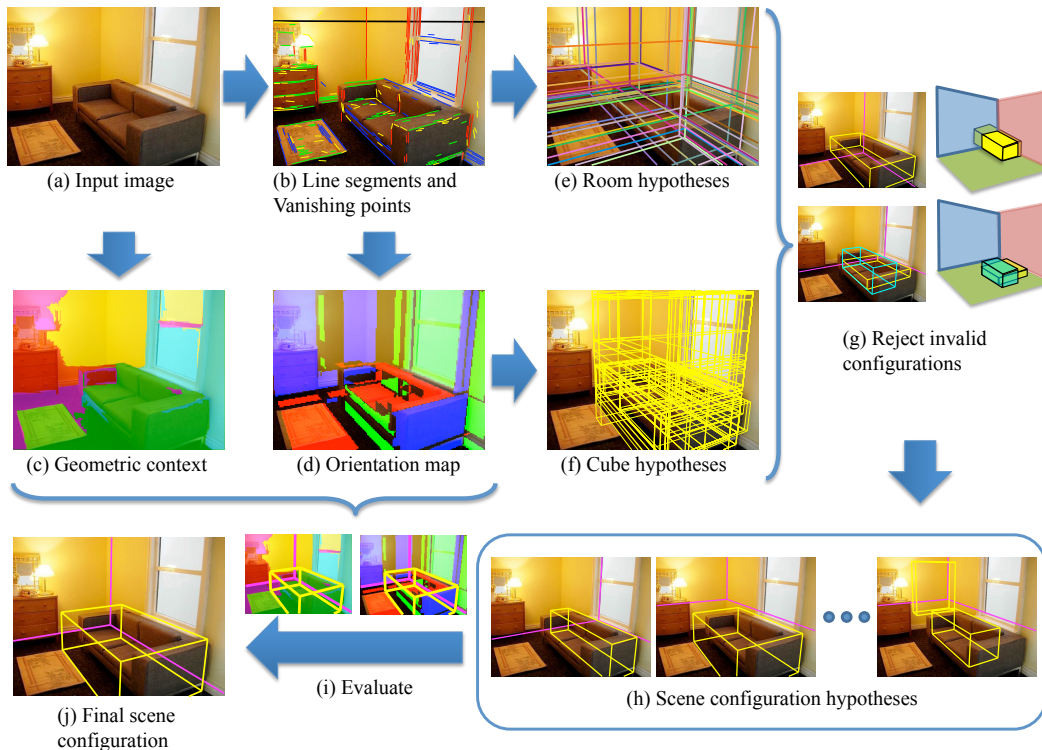

(a) Input image

(b) Line segments and Vanishing points

(e) Room hypotheses

(c) Geometric context

(d) Orientation map

(f) Cube hypotheses

(g) Reject invalid configurations

(i) Evaluate

(h) Scene configuration hypotheses

(j) Final scene configuration

Figure 2: Overview of our approach for estimating the spatial layout of the room and the objects.

volumetric constraints. The estimated surface geometry is therefore used as features in a scoring function that evaluates a given scene configuration.

For estimating surface geometry we use two methods: the line-sweeping algorithm [11] and a multiple segmentation classifier [8]. The line-sweeping algorithm takes line segments as input and predicts an orientation map in which regions are classified as surfaces into one of the three possible orientations. Figure 2(d) shows an example of an orientation map. The region estimated as horizontal surface is colored in red, and vertical surfaces are colored in green and blue, corresponding to the associated vanishing point. This orientation map is used to evaluate scene configuration hypotheses. The multiple segmentation classifier [8] takes the full image as input, uses image features, such as combinations of color and texture, and predicts geometric context represented by surface geometry labels for each superpixel (floor, ceiling, vertical (left, center, right), solid, and porous regions). Similar to orientation maps, the predicted labels are used to evaluate scene configuration hypotheses.

## 4 Generating Scene Configuration Hypothesis

Given the local surface geometry and the oriented line segments extracted from the image, we now create multiple hypotheses for possible spatial layout of the room and object layout in the room. These hypotheses are then combined to produce scene configuration layout such that all the objects occupy exclusive 3D volumes and the objects are inside the freespace of the room defined by the walls.

### 4.1 Generating Room Hypotheses

A room hypothesis encodes the position and orientation of walls, floor, and ceiling. In this paper, we represent a room hypothesis by a parametric box model [12]. Room hypotheses are generated from line segments in a way similar to the method described in Lee *et al.* [11]. They examine exhaustive combinations of line segments and check which of the resulting combinations define physically valid room models. Instead, we sample random tuples of line segments lines that define the boundaries of the parametric box. Only the minimum number of line segments to define the parametric room model are sampled. Figure 2(e) shows examples of generated room hypotheses.

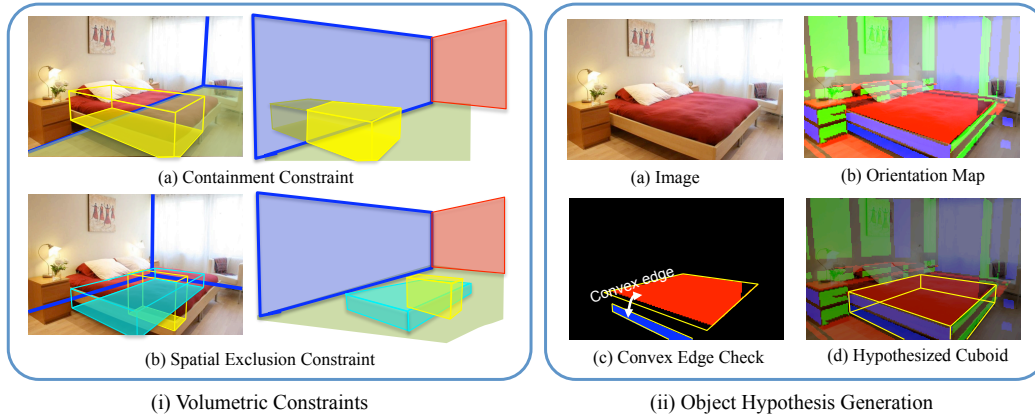

(a) Containment Constraint

(a) Image     (b) Orientation Map

(b) Spatial Exclusion Constraint

(c) Convex Edge Check     (d) Hypothesized Cuboid

(i) Volumetric Constraints     (ii) Object Hypothesis Generation

Figure 3: (i) Examples of volumetric constraint violation. (ii) Object hypothesis generation: we use the orientation maps to generate object hypotheses by finding convex edges.

## 4.2 Generating Object Hypotheses

Our goal is to extract the 3D geometry of the clutter objects to perform 3D spatial reasoning. Estimating precise 3D models of objects from a single image is an extremely difficult problem and probably requires recognition of object classes such as couches and tables. However, our goal is to perform coarse 3D reasoning about the spatial layout of rooms and spatial layout of objects in the room. We only need to model a subset of objects in the scene to provide enough constraints for volumetric reasoning. Therefore, we adopt a coarse 3D model of objects in the scene and model each object-volume as cuboids. We found that parameterizing objects as cuboids provides a good approximation to the occupied volume in man-made environments. Furthermore, by modeling objects by a parametric model of a cuboid, we can determine the location and dimensions in 3D up to scale, which allows volumetric reasoning about the 3D interaction between objects and the room.

We generate object hypotheses from the orientation map described above. Figure 3(ii)(a)(b) shows an example scene and its orientation map. The three colors represent the three possible plane orientations used in the orientation map. We can see from the figure that the distribution of surfaces on the objects estimated by the orientation map suggests the presence of a cuboidal object. Figure 3(ii)(c) shows a pair of regions which can potentially form a convex edge if the regions represent the visible surfaces on a cuboidal object.

We test all pairs of regions in the orientation map to check whether they can form convex edges. This is achieved by checking the estimated orientation of the regions and the spatial location of the regions with respect to the vanishing points. If the region pair can form a convex corner, we utilize these regions to form an object hypothesis. To generate a cuboidal object hypothesis from pairs of regions, we first fit tight bounding quadrilaterals (Figure 3(ii)(c)) to each region in the pair and then sample all combinations of three points out of the eight vertices on the two quadrilaterals, which do not lie on a plane. Three is the minimum number of points (with $(x, y)$ coordinates) that have enough information to define a cuboid projected onto a 2D image plane, which has five degrees of freedom. We can then hypothesize a cuboid, whose corner best apprximates the three points. Figure 3(ii)(d) shows a sample of a cuboidal object hypothesis generated from the given orientation map.

## 4.3 Volumetric Compatibility of Scene Configuration

Given a room configuration and a set of candidate objects, a key operation is to evaluate whether the resulting combination satisfies the three fundamental volumetric compatibility constraints described in Section 2. The problem of estimating the three dimensional layout of a scene from a single image is inherently ambiguous because any measurement from a single image can only be determined up to scale. In order to test the volumetric compatibility of room-object hypotheses pairs and object-object hypotheses pairs, we make the assumption that all objects rest on the floor. This assumption fixes the scale ambiguity between room and object hypotheses and allows us to reason about their 3D location.

To test whether an object is contained within the free space of a room, we check whether the projection of the bottom surface of the object onto the image is completely contained within the projection of the floor surface of the room. If the projection of the bottom surface of the object is not completely

within the floor surface, the corresponding 3D object model must be protruding into the walls of the room. Figure 3(i)(a) shows an example of an incompatible room-object pair.

Similarly, to test whether the volume occupied by two objects is exclusive, we assume that the two objects rest on the same floor plane and we compare the projection of their bottom surfaces onto the image. If there is any overlap between the projections of the bottom surface of the two object hypotheses, that means that they occupy intersecting volumes in 3D. Figure 3(i)(b) shows an example of an incompatible object-object pair.

## 5 Evaluating Scene Configurations

### 5.1 Inference

Given an image $x$, a set of room hypotheses $\{r_1, r_2, ..., r_n\}$, and a set of object hypotheses $\{o_1, o_2, ..., o_m\}$, our goal is to find the best scene configuration $\mathbf{y} = (\mathbf{y_r}, \mathbf{y_o})$, where $\mathbf{y_r} = (y_r^1, ..., y_r^n)$, $\mathbf{y_o} = (y_o^1, ..., y_o^m)$. $y_r^i = 1$ if room hypothesis $r_i$ is used in the scene configuration and $y_r^i = 0$ otherwise, and $y_o^i = 1$ if object hypothesis $o_i$ is present in the scene configuration and $y_o^i = 0$ otherwise. Note that $\sum_i y_r^i = 1$ as only one room hypothesis is needed to define the scene configuration.

Suppose that we are given a function $f(x, \mathbf{y})$ that returns a score for $\mathbf{y}$. Finding the best scene configuration $\mathbf{y}^* = \arg\max_{\mathbf{y}} f(x, \mathbf{y})$ through testing all possible scene configurations requires $n \cdot 2^m$ evaluations of the score function. We resort to using beam search (fixed width search tree) to keep the computation manageable by avoiding evaluating all scene configurations.

In the first level of the search tree, scene configurations with a room hypothesis and no object hypothesis are evaluated. In the following levels, an object hypothesis is added to its parent configuration and the configuration is evaluated. The top $k_l$ nodes with the highest score are added to the search tree as the child node, where $k_l$ is a pre-determined beam width for level $l$.[2] The search is continued for a fixed number of levels or until no cubes that are compatible with existing configurations can be added. After the search tree has been explored, the best scoring node in the tree is returned as the best scene configuration.

### 5.2 Learning the Score Function

We set the score function to $f(x, \mathbf{y}) = w^T \psi(x, \mathbf{y}) + w_\phi^T \phi(\mathbf{y})$, where $\psi(x, \mathbf{y})$ is a feature vector for a given image $x$ and measures the compatibility of the scene configuration $y$ with the estimated surface geometry. $\phi(\mathbf{y})$ is the penalty term for incompatible configurations and penalizes the room and object configurations which violate volumetric constraints.

We use structured SVM [16] to learn the weight vector $w$. The weights are learned by solving

$$\min_{w, \xi} \frac{1}{2} \|w\|^2 + C \sum_i \xi_i$$

$$s.t. \ w^T \psi(x_i, \mathbf{y_i}) - w^T \psi(x_i, \mathbf{y}) - w_\phi^T \phi(\mathbf{y}) \geq \Delta(\mathbf{y_i}, \mathbf{y}) - \xi_i, \forall i, \forall \mathbf{y}$$

$$\xi_i \geq 0, \forall i,$$

where $x_i$ are images, $\mathbf{y_i}$ are the ground truth configuration, $\xi_i$ are slack variables, and $\Delta(\mathbf{y_i}, \mathbf{y})$ is the loss function that measures the error of configuration $\mathbf{y}$. Tsochantaridis [16] deals with the large number of constraints by iteratively adding the most violated constraints. We simplify this by sampling a fixed number of configurations per each training image, using the same beam search process used for inference, and solving using quadratic programming.

**Loss Function:** The loss function $\Delta(\mathbf{y_i}, \mathbf{y})$ is the percentage of pixels in the entire image having incorrect label. For example, pixels that are labeled as left wall when they actually belong to the center wall, or pixels labeled as object when they actually belong to the floor would be counted as incorrectly labeled pixels. A wall is labeled as center if the surface normal is within 45 degrees from the camera optical axis and labeled as left or right, otherwise.

**Feature Vector:** The feature vector $\psi(x, \mathbf{y})$ is computed by measuring how well each surface in the scene configuration $\mathbf{y}$ is supported by the orientation map and the geometric context. A feature

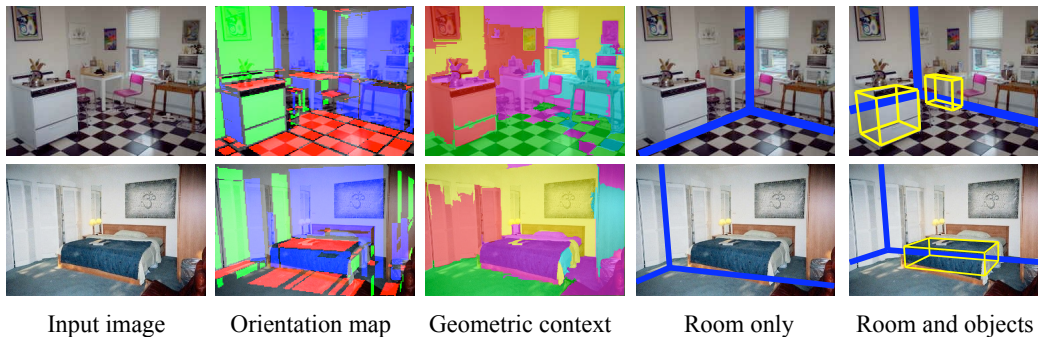

| | Input image | Orientation map | Geometric context | Room only | Room and objects |

Figure 4: Two qualitative examples showing how 3D volumetric reasoning aids estimation of the spatial layout of the room.

|  | OM+GC | OM | GC |
|---|---|---|---|
| No object reasoning | 18.6% | 24.7% | 22.7% |
| Volumetric reasoning | 16.2% | 19.5% | 20.2% |

Table 1: Percentage of pixels with correct estimate of room surfaces. First row performs no reasoning about objects. Second row is our approach with 3D volumetric reasoning of objects. Columns shows the features that are used. OM: Orientation map from [11]. GC: Geometric context from [8].

is computed for each of the six surfaces in the scene configuration (floor, left wall, center wall, right wall, ceiling, object) as the relative area which the orientation map or the geometric context correctly explains the attribute of the surface. This results in a twelve dimensional feature vector for a given scene configuration. For example, the feature for the floor surface in the scene configuration is computed by the relative area which the orientation map predicts a horizontal surface, and the area which the geometric context predicts a floor label.

**Volumetric Penalty:** The penalty term $\phi(y)$ measures how much the volumetric constraints are violated. (1) The first term $\phi(y_r, y_o)$ measures the volumetric intersection between the volume defined by room walls and objects. It penalizes the configurations where the object hypothesis lie outside the room volume and the penalty is proportional to the volume outside the room. (2) The second term $\sum_{i,j} \phi(y_o^i, y_o^j)$ measures the volume intersection between two objects $(i,j)$. This penalty from this term is proportional to the overlap of the cubes projected on the floor.

## 6  Experimental Results

We evaluated our 3D geometric reasoning approach on an indoor image dataset introduced in [12]. The dataset consists of 314 images, and the ground-truth consists of the marked spatial layout of the room and the clutter layouts. For our experiments, we use the same training-test split as used in [12] (209 training and 105 test images). We use training images to estimate the weight vector.

**Qualitative Evaluation:** Figure 4 illustrates the benefit of 3D spatial reasoning introduced in our approach. If no 3D clutter reasoning is used and the room box is fitted to the orientation map and geometric context, the box gets fit to the object surfaces and therefore leads to substantial error in the spatial layout estimation. However, if we use 3D object reasoning walls get pushed due to the containment constraint and the spatial layout estimation improves. We can also see from the examples that extracting a subset of objects in the scene is enough for reasoning and improving the spatial layout estimation. Figure 5 and 6 shows more examples of the spatial layout and the estimated clutter objects in the images. Additional results are in the supplementary material.

**Quantitative Evaluation:** We evaluate the performance of our approach in estimating the spatial layout of the room. We use the pixel-based measure introduced in [12] which counts the percentage of pixels on the room surfaces that disagree with the ground truth. For comparison, we employ the simple multiple segmentation classifier [8] and the recent approach introduced in [12] as baselines. The images in the dataset have significant clutter; therefore, simple classification based approaches with no clutter reasoning perform poorly and have an error of $26.5\%$. The state-of-the-art approach [12] which utilizes clutter reasoning in the image plane has an error of $21.2\%$. On the other hand, our

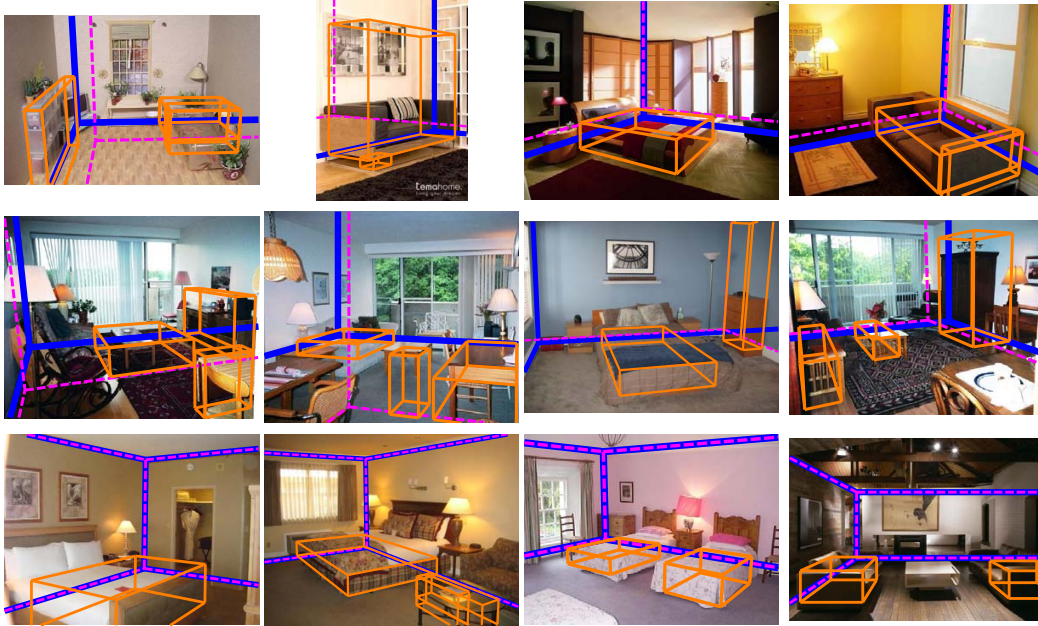

Figure 5: Additional examples to show the performance on a wide variety of scenes. Dotted lines represent the room estimate without object reasoning.

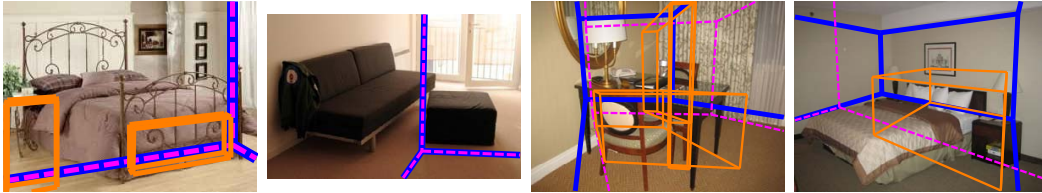

Figure 6: Failure examples. The first two examples are the failure cases when the cuboids are either missed or estimated wrong. The last two failure cases are due to errors in vanishing point estimation.

approach which uses a parametric model of clutter and simple 3D volumetric reasoning outperforms both the approaches and has an error of $16.2\%$.

We also performed several experiments to measure the significance of each step and features in our approach. When we only use the surface layout estimates from [8] as features of the cost function, our approach has an error rate of $20.2\%$ whereas using only orientation maps as features yields an error rate of $19.5\%$. We also tried several search techniques to search the space of hypotheses. With a greedy approach (best cube added at each iteration) to search the hypothesis space, we achieved an error rate of $19.2\%$, which shows that early commitment to partial configurations leads to error and search strategy that allows late commitment, such as beam search, should be used.

## 7  Conclusion

In this paper, we have proposed the use of volumetric reasoning between objects and surfaces of room layout to recover the spatial layout of a scene. By parametrically representing the 3D volume of objects and rooms, we can apply constraints for volumetric reasoning, such as spatial exclusion and containment. Our experiments show that volumetric reasoning improves the estimate of the room layout and provides a richer interpretation about objects in the scene. The rich geometric information provided by our method can provide crucial information for object recognition and eventually aid in complete scene understanding.

## 8  Acknowledgements

This research was supported by NSF Grant EEEC-0540865, ONR MURI Grant N00014-07-1-0747, NSF Grant IIS-0905402, and ONR Grant N000141010766.

## Footnotes

[1] $O(n \cdot 2^m)$ where $n$ is the number of room hypotheses and $m$ is the number of object hypotheses

[2]We set $k_l$ to $(100, 5, 2, 1)$, with a maximum of 4 levels. The results were not sensitive to these parameters.

# References

[1] L. Roberts. Machine perception of 3-d solids. In: PhD. Thesis. (1965)

[2] A. Guzman. Decomposition of a visual scene into three dimensional bodies. In Proceedings of Fall Joint Computer Conference, 1968.

[3] D. A. Waltz. Generating semantic descriptions from line drawings of scenes with shadows. Technical report, MIT, 1972.

[4] J. Coughlan, and A. Yuille. Manhattan world: Compass direction from a single image by bayesian inference. In proc. ICCV, 1999.

[5] J. Kosecka, and W. Zhang. Video Compass. In proc. ECCV, 2002.

[6] J. Kosecka, and W. Zhang. Extraction, matching, and pose recovery based on dominant rectangular structures. CVIU, 2005.

[7] S. Yu, H. Zhang, and J. Malik. Inferring Spatial Layout from A Single Image via Depth-Ordered Grouping. IEEE Computer Society Workshop on Perceptual Organization in Computer Vision, 2008

[8] D. Hoiem, A. Efros, and M. Hebert. Recovering surface layout from an image. IJCV, 75(1), 2007.

[9] A. Saxena, M. Sun, and A. Ng. Make3d: Learning 3D scene structure from a single image. PAMI, 2008.

[10] E. Delage, H. Lee, and A. Ng. A dynamic bayesian network model for autonomous 3D reconstruction from a single indoor image. CVPR, 2006.

[11] D. Lee, M. Hebert, and T. Kanade. Geometric reasoning for single image structure recovery. In proc. CVPR, 2009.

[12] V. Hedau, D. Hoiem, and D. Forsyth. Recovering the spatial layout of cluttered rooms. In proc. ICCV, 2009.

[13] H. Wang, S. Gould, and D. Koller, Discriminative Learning with Latent Variables for Cluttered Indoor Scene Understanding. ECCV, 2010.

[14] V. Hedau, D. Hoiem, and D. Forsyth. Thinking Inside the Box: Using Appearance Models and Context Based on Room Geometry. ECCV, 2010.

[15] A. Gupta, A. Efros, and M. Hebert. Blocks World Revisited: Image Understanding using Qualitative Geometry and Mechanics. ECCV, 2010.

[16] I. Tsochantaridis, T. Joachims, T. Hofmann and Y. Altun: Large Margin Methods for Structured and Interdependent Output Variables, JMLR, Vol. 6, pages 1453-1484, 2005

